# Exploring Unknown Environments with Real-Time Search or Reinforcement Learning

**Sven Koenig**
College of Computing, Georgia Institute of Technology
skoenig@cc.gatech.edu

## Abstract

Learning Real-Time A* (LRTA*) is a popular control method that interleaves planning and plan execution and has been shown to solve search problems in known environments efficiently. In this paper, we apply LRTA* to the problem of getting to a given goal location in an initially unknown environment. Uninformed LRTA* with maximal lookahead always moves on a shortest path to the closest unvisited state, that is, to the closest potential goal state. This was believed to be a good exploration heuristic, but we show that it does not minimize the worst-case plan-execution time compared to other uninformed exploration methods. This result is also of interest to reinforcement-learning researchers since many reinforcement learning methods use asynchronous dynamic programming, interleave planning and plan execution, and exhibit optimism in the face of uncertainty, just like LRTA*.

## 1 Introduction

Real-time (heuristic) search methods are domain-independent control methods that interleave planning and plan execution. They are based on agent-centered search [Dasgupta *et al.*, 1994; Koenig, 1996], which restricts the search to a small part of the environment that can be reached from the current state of the agent with a small number of action executions. This is the part of the environment that is immediately relevant for the agent in its current situation. The most popular real-time search method is probably the Learning Real-Time A* (LRTA*) method [Korf, 1990]. It has a solid theoretical foundation and the following advantageous properties: First, it allows for fine-grained control over how much planning to do between plan executions and thus is an any-time contract algorithm [Russell and Zilberstein, 1991]. Second, it can use heuristic knowledge to guide planning, which reduces planning time without sacrificing solution quality. Third, it can be interrupted at any state and resume execution at a different state. Fourth, it amortizes learning over several search episodes, which allows it to find plans with suboptimal plan-execution time fast and then improve the plan-execution time as it solves similar planning tasks, until its plan-execution time is optimal. Thus, LRTA* always has a small sum of planning and plan-execution

Initially, $u(s) = 0$ for all $s \in S$.

1. $s_{current} := s_{start}$.
2. If $s_{current} \in G$, then stop successfully.
3. Generate a local search space $S_{lss} \subseteq S$ with $s_{current} \in S_{lss}$ and $S_{lss} \cap G = \emptyset$.
4. Update $u(s)$ for all $s \in S_{lss}$ (Figure 2).
5. $a := $ one-of arg $\min_{a \in A(s_{current})} u(succ(s_{current}, a))$.
6. Execute action $a$.
7. $s_{current} := succ(s_{current}, a)$.
8. If $s_{current} \in S_{lss}$, then go to 5.
9. Go to 2.

Figure 1: Uninformed LRTA*

1. For all $s \in S_{lss}$: $u(s) := \infty$.
2. If $u(s) < \infty$ for all $s \in S_{lss}$, then return.
3. $s' := $ one-of arg $\min_{s \in S_{lss}: u(s) = \infty} \min_{a \in A(s)} u(succ(s, a))$.
4. If $\min_{a \in A(s')} u(succ(s', a)) = \infty$, then return.
5. $u(s') := 1 + \min_{a \in A(s')} u(succ(s', a))$.
6. Go to 2.

Figure 2: Value-Update Step

time, and it minimizes the plan-execution time in the long run in case similar planning tasks unexpectedly repeat. This is important since no search method that executes actions before it has solved a planning task completely can guarantee to minimize the plan-execution time right away.

Real-time search methods have been shown to be efficient alternatives to traditional search methods in known environments. In this paper, we investigate real-time search methods in unknown environments. In such environments, real-time search methods allow agents to gather information early. This information can then be used to resolve some of the uncertainty and thus reduce the amount of planning done for unencountered situations.

We study robot-exploration tasks without actuator and sensor uncertainty, where the sensors on-board the robot can uniquely identify its location and the neighboring locations. The robot does not know the map in advance, and thus has to explore its environment sufficiently to find the goal and a path to it. A variety of methods can solve these tasks, including LRTA*. The proceedings of the AAAI-97 Workshop on On-Line Search [Koenig *et al.*, 1997] give a good overview of some of these techniques. In this paper, we study whether uninformed LRTA* is able to minimize the worst-case plan-execution time over all state spaces with the same number of states provided that its lookahead is sufficiently large. Uninformed LRTA* with maximal lookahead always moves on a shortest path to the closest unvisited state, that is, to the closest potential goal state – it exhibits optimism in the face of uncertainty [Moore and Atkeson, 1993]. We show that this exploration heuristic is not as good as it was believed to be. This solves the central problem left open in [Pemberton and Korf, 1992] and improves our understanding of LRTA*. Our results also apply to learning control for tasks other than robot exploration, for example the control tasks studied in [Davies *et al.*, 1998]. They are also of interest to reinforcement-learning researchers since many reinforcement learning methods use asynchronous dynamic programming, interleave planning and plan execution, and exhibit optimism in the face of uncertainty, just like LRTA* [Barto *et al.*, 1995; Kearns and Singh, 1998].

## 2  LRTA*

We use the following notation to describe LRTA*: $S$ denotes the finite set of states of the environment, $s_{start} \in S$ the start state, and $\emptyset \neq G \subseteq S$ the set of goal states. The number of states is $n := |S|$. $A(s) \neq \emptyset$ is the finite, nonempty set of actions that can be executed in state $s \in S$. $succ(s, a)$ denotes the successor state that results from the execution of action $a \in A(s)$ in state $s \in S$. We also use two operators with the following semantics: Given

a set $X$, the expression "one-of $X$" returns an element of $X$ according to an arbitrary rule. A subsequent invocation of "one-of $X$" can return the same or a different element. The expression "arg $\min_{x \in X} f(x)$" returns the elements $x \in X$ that minimize $f(x)$, that is, the set $\{x \in X | f(x) = \min_{x' \in X} f(x')\}$.

We model environments (topological maps) as state spaces that correspond to undirected graphs, and assume that it is indeed possible to reach a goal state from the start state. We measure the distances and thus plan-execution time in action executions, which is reasonable if every action can be executed in about the same amount of time. The graph is initially unknown. The robot can always observe whether its current state is a goal state, how many actions can be executed in it, and which successor states they lead to but not whether the successor states are goal states. Furthermore, the robot can identify the successor states when it observes them again at a later point in time. This assumption is realistic, for example, if the states look sufficiently different or the robot has a global positioning system (GPS) available.

LRTA* learns a map of the environment and thus needs memory proportional to the number of states and actions observed. It associates a small amount of information with the states in its map. In particular, it associates a *u-value* $u(s)$ with each state $s \in S$. The u-values approximate the goal distances of the states. They are updated as the search progresses and used to determine which actions to execute. Figure 1 describes LRTA*: LRTA* first checks whether it has already reached a goal state and thus can terminate successfully (Line 2). If not, it generates the local search space $S_{lss} \subseteq S$ (Line 3). While we require only that the current state is part of the local search space and the goal states are not [Barto *et al.*, 1995], in practice LRTA* constructs $S_{lss}$ by searching forward from the current state. LRTA* then updates the u-values of all states in the local search space (Line 4), as shown in Figure 2. The value-update step assigns each state its goal distance under the assumption that the u-values of all states outside of the local search space correspond to their correct goal distances. Formally, if $u(s) \in [0, \infty]$ denotes the u-values before the value-update step and $\bar{u}(s) \in [0, \infty]$ denotes the u-values afterwards, then $\bar{u}(s) = 1 + \min_{a \in A(s)} \bar{u}(succ(s, a))$ for all $s \in S_{lss}$ and $\bar{u}(s) = u(s)$ otherwise. Based on these u-values, LRTA* decides which action to execute next (Line 5). It greedily chooses the action that minimizes the u-value of the successor state (ties are broken arbitrarily) because the u-values approximate the goal distances and LRTA* attempts to decrease its goal distance as much as possible. Finally, LRTA* executes the selected action (Line 6) and updates its current state (Line 7). Then, if the new state is still part of the local search space used previously, LRTA* selects another action for execution based on the current u-values (Line 8). Otherwise, it iterates (Line 9). (The behavior of LRTA* with either minimal or maximal lookahead does not change if Line 8 is deleted.)

## 3 Plan-Execution Time of LRTA* for Exploration

In this section, we study the behavior of LRTA* with minimal and maximal lookaheads in unknown environments. We assume that no a-priori heuristic knowledge is available and, thus, that LRTA* is uninformed. In this case, the u-values of all unvisited states are zero and do not need to be maintained explicitly.

**Minimal Lookahead:** The lookahead of LRTA* is minimal if the local search space contains only the current state. LRTA* with minimal lookahead performs almost no planning between plan executions. Its behavior in initially known and unknown environments is identical. Figure 3 shows an example.

Let $gd(s)$ denote the goal distance of state $s$. Then, according to one of our previous results, uninformed LRTA* with any lookahead reaches a goal state after at most $\sum_{s \in S} gd(s)$ action executions [Koenig and Simmons, 1995]. Since $\sum_{s \in S} gd(s) \leq \sum_{i=0}^{n-1} i = 1/2n^2 - 1/2n$,

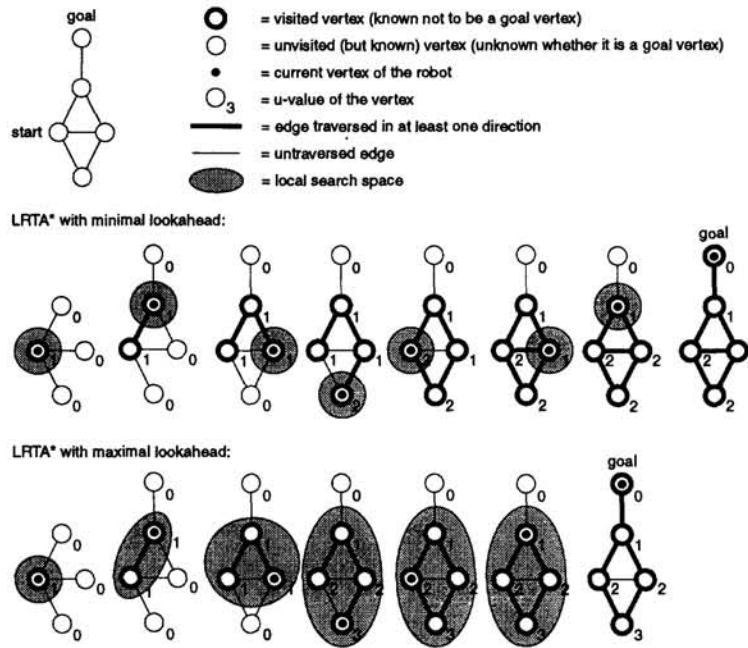

Figure 3: Example

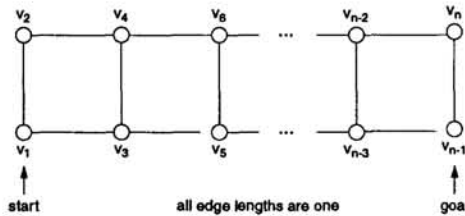

Figure 4: A Planar Undirected Graph

uninformed LRTA* with any lookahead reaches a goal state after $O(n^2)$ action executions.

This upper bound on the plan-execution time is tight in the worst case for uninformed LRTA* with minimal lookahead, even if the number of actions that can be executed in any state is bounded from above by a small constant (here: three). Figure 4, for example, shows a rectangular grid-world for which uninformed LRTA* with minimal lookahead reaches a goal state in the worst case only after $\Theta(n^2)$ action executions. In particular, LRTA* can traverse the state sequence that is printed by the following program in pseudo code. The scope of the for-statements is shown by indentation.

```
for i := n-3 downto n/2 step 2
  for j := 1 to i step 2
    print j
  for j := i+1 downto 2 step 2
    print j
for i := 1 to n-1 step 2
  print i
```

In this case, LRTA* executes $3n^2/16 - 3/4$ actions before it reaches the goal state (for $n \geq 2$ with $n \bmod 4 = 2$). For example, for $n = 10$, it traverses the state sequence $s_1$, $s_3$, $s_5$, $s_7$, $s_8$, $s_6$, $s_4$, $s_2$, $s_1$, $s_3$, $s_5$, $s_6$, $s_4$, $s_2$, $s_1$, $s_3$, $s_5$, $s_7$, and $s_9$.

**Maximal Lookahead:** As we increase the lookahead of LRTA*, we expect that its plan-execution time tends to decrease because LRTA* uses more information to decide which

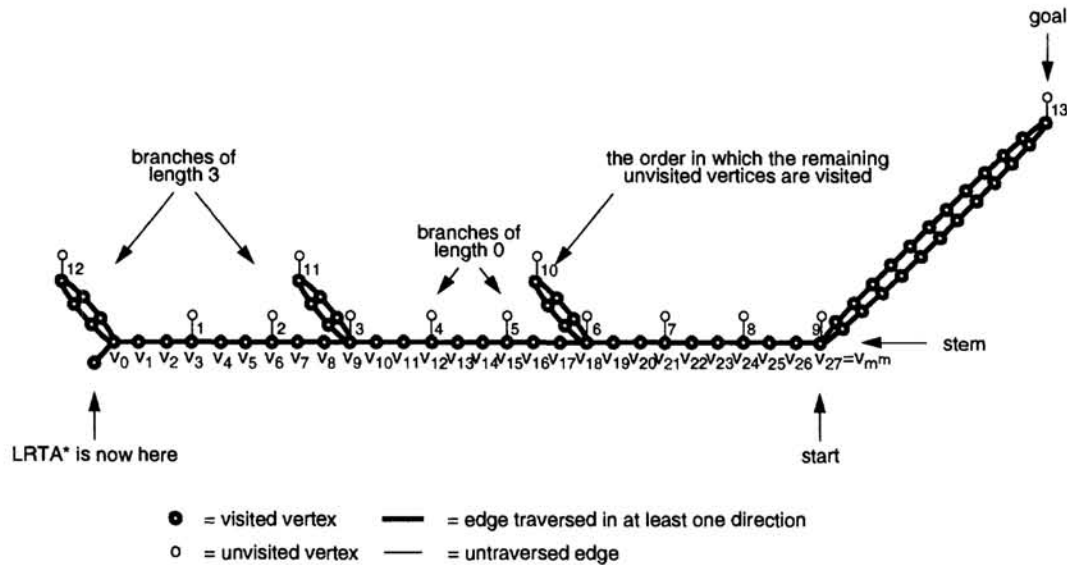

Figure 5: Another Planar Undirected Graph ($m = 3$)

action to execute next. This makes it interesting to study LRTA* with maximal lookahead.

The lookahead of LRTA* is maximal in known environments if the local search space contains all non-goal states. In this case, LRTA* performs a complete search without interleaving planning and plan execution and follows a shortest path from the start state to a closest goal state. Thus, it needs $gd(s_{start})$ action executions. No other method can do better than that.

The maximal lookahead of LRTA* is necessarily smaller in initially unknown environments than in known environments because its value-update step can only search the known part of the environment. Therefore, the lookahead of LRTA* is maximal in unknown environments if the local search space contains all *visited* non-goal states. Figure 3 shows an example.

Uninformed LRTA* with maximal lookahead always moves on a shortest path to the closest unvisited state, that is, to the closest potential goal state. This appears to be a good exploration heuristic. [Pemberton and Korf, 1992] call this behavior "incremental best-first search," but were not able to prove or disprove whether this locally optimal search strategy is also globally optimal. Since this exploration heuristic has been used on real mobile robots [Thrun *et al.*, 1998], we study how well its plan-execution time compares to the plan-execution time of other uninformed exploration methods. We show that the worst-case plan-execution time of uninformed LRTA* with maximal lookahead in unknown environments is $\Omega(\frac{\log n}{\log \log n} n)$ action executions and thus grows faster than linearly in the number of states $n$. It follows that the plan-execution time of LRTA* is not optimal in the worst case, since depth-first search needs a number of action executions in the worst case that grows only linearly in the number of states.

Consider the graph shown in Figure 5, that is a variation of a graph in [Koenig and Smirnov, 1996]. It consists of a stem with several branches. Each branch consists of two parallel paths of the same length that connect the stem to a single edge. The length of the branch is the length of each of the two paths. The stem has length $m^m$ for some integer $m \geq 3$ and consists of the vertices $v_0, v_1, \ldots, v_{m^m}$. For each integer $i$ with $1 \leq i \leq m$ there are $m^{m-i}$ branches of length $\sum_{j=1}^{i-1} m^j$ each (including branches of length zero). These branches attach to the stem at the vertices $v_{j\,m^i}$ for integers $j$; if $i$ is even, then $0 \leq j \leq m^{m-i} - 1$, otherwise $1 \leq j \leq m^{m-i}$. There is one additional single edge that attaches to vertex $v_0$.

$v_{m^m}$ is the starting vertex. The vertex at the end of the single edge of the longest branch is the goal vertex. Notice that the graph is planar. This is a desirable property since non-planar graphs are, in general, rather unrealistic models of maps.

Uninformed LRTA* with maximal lookahead can traverse the stem repeatedly forward and backward, and the resulting plan-execution time is large compared to the number of vertices that are necessary to mislead LRTA* into this behavior. In particular, LRTA* can behave as follows: It starts at vertex $v_{m^m}$ and traverses the whole stem and all branches, excluding the single edges at their end, and finally traverses the additional edge attached to vertex $v_0$, as shown in Figure 5. At this point, LRTA* knows all vertices. It then traverses the whole stem, visiting the vertices at the ends of the single edges of the branches of length 0. It then switches directions and travels along the whole stem in the opposite direction, this time visiting the vertices at the end of the single edges of the branches of length $m$, and so forth, switching directions repeatedly. It succeeds when it finally uses the longest branch and discovers the goal vertex. To summarize, the vertices at the ends of the branches are tried out in the order indicated in Figure 5. The total number of edge traversals is $\Omega(m^{m+1})$ since the stem of length $m^m$ is traversed $m + 1$ times. To be precise, the total number of edge traversals is $(m^{m+3} + 3m^{m+2} - 8m^{m+1} + 2m^2 - m + 3)/(m^2 - 2m + 1)$. It holds that $n = \Theta(m^m)$ since $n = (3m^{m+2} - 5m^{m+1} - m^m + m^{m-1} + 2m^2 - 2m + 2)/(m^2 - 2m + 1)$. This implies that $m = \Omega(\frac{\log n}{\log \log n})$ since it holds that, for $k > 1$ and all sufficiently large $m$ (to be precise: $m$ with $m \geq k$)

$$\frac{\log_k(m^m)}{\log_k \log_k(m^m)} = \frac{1}{\frac{\log_k \log_k(m^m)}{\log_k(m^m)}} = \frac{1}{\frac{\log_k m + \log_k \log_k m}{m \log_k m}} = \frac{1}{\frac{1}{m} + \frac{\log_k \log_k m}{m \log_k m}} \leq \frac{1}{\frac{1}{m} + 0} = m.$$

Put together, it follows that the total number of edge traversals is $\Omega(m^{m+1}) = \Omega(m\,n) = \Omega(\frac{\log n}{\log \log n}\,n)$. (We also performed a simulation that confirmed our theoretical results.)

The graph from Figure 5 can be modified to cause LRTA* to behave similarly even if the assumptions of the capabilities of the robot or the environment vary from our assumptions here, including the case where the robot can observe only the actions that lead to unvisited states but not the states themselves.

## 4   Future Work

Our example provided a lower bound on the plan-execution time of uninformed LRTA* with maximal lookahead in unknown environments. The lower bound is barely super-linear in the number of states. A tight bound is currently unknown, although upper bounds are known. A trivial upper bound, for example, is $O(n^2)$ since LRTA* executes at most $n - 1$ actions before it visits another state that it has not visited before and there are only $n$ states to visit. A tighter upper bound follows directly from [Koenig and Smirnov, 1996]. It was surprisingly difficult to construct our example. It is currently unknown, and therefore a topic of future research, for which classes of graphs the worst-case plan-execution time of LRTA* is optimal up to a constant factor and whether these classes of graphs correspond to interesting and realistic environments. It is also currently unknown how the bounds change as LRTA* becomes more informed about where the goal states are.

## 5   Conclusions

Our work provides a first analysis of uninformed LRTA* in unknown environments. We studied versions of LRTA* with minimal and maximal lookaheads and showed that their

worst-case plan-execution time is not optimal, not even up to a constant factor. The worst-case plan-execution time of depth-first search, for example, is smaller than that of LRTA* with either minimal or maximal lookahead. This is not to say that one should always prefer depth-first search over LRTA* since, for example, LRTA* can use heuristic knowledge to direct its search towards the goal states. LRTA* can also be interrupted at any location and get restarted at a different location. If the batteries of the robot need to get recharged during exploration, for instance, LRTA* can be interrupted and later get restarted at the charging station. While depth-first search could be modified to have these properties as well, it would lose some of its simplicity.

## Acknowledgments

Thanks to Yury Smirnov for our collaboration on previous work which this paper extends. Thanks also to the reviewers for their suggestions for improvements and future research directions. Unfortunately, space limitations prevented us from implementing all of their suggestions in this paper.

## References

(Barto *et al.*, 1995) Barto, A.; Bradtke, S.; and Singh, S. 1995. Learning to act using real-time dynamic programming. *Artificial Intelligence* 73(1):81–138.

(Dasgupta *et al.*, 1994) Dasgupta, P.; Chakrabarti, P.; and DeSarkar, S. 1994. Agent searching in a tree and the optimality of iterative deepening. *Artificial Intelligence* 71:195–208.

(Davies *et al.*, 1998) Davies, S.; Ng, A.; and Moore, A. 1998. Applying online search techniques to reinforcement learning. In *Proceedings of the National Conference on Artificial Intelligence.* 753–760.

(Kearns and Singh, 1998) Kearns, M. and Singh, S. 1998. Near-optimal reinforcement learning in polynomial time. In *Proceedings of the International Conference on Machine Learning.* 260–268.

(Koenig and Simmons, 1995) Koenig, S. and Simmons, R.G. 1995. Real-time search in non-deterministic domains. In *Proceedings of the International Joint Conference on Artificial Intelligence.* 1660–1667.

(Koenig and Smirnov, 1996) Koenig, S. and Smirnov, Y. 1996. Graph learning with a nearest neighbor approach. In *Proceedings of the Conference on Computational Learning Theory.* 19–28.

(Koenig *et al.*, 1997) Koenig, S.; Blum, A.; Ishida, T.; and Korf, R., editors 1997. *Proceedings of the AAAI-97 Workshop on On-Line Search.* AAAI Press.

(Koenig, 1996) Koenig, S. 1996. Agent-centered search: Situated search with small look-ahead. In *Proceedings of the National Conference on Artificial Intelligence.* 1365.

(Korf, 1990) Korf, R. 1990. Real-time heuristic search. *Artificial Intelligence* 42(2-3):189–211.

(Moore and Atkeson, 1993) Moore, A. and Atkeson, C. 1993. Prioritized sweeping: Reinforcement learning with less data and less time. *Machine Learning* 13:103–130.

(Pemberton and Korf, 1992) Pemberton, J. and Korf, R. 1992. Incremental path planning on graphs with cycles. In *Proceedings of the International Conference on Artificial Intelligence Planning Systems.* 179–188.

(Russell and Zilberstein, 1991) Russell, S. and Zilberstein, S. 1991. Composing real-time systems. In *Proceedings of the International Joint Conference on Artificial Intelligence.* 212–217.

(Thrun *et al.*, 1998) Thrun, S.; Bücken, A.; Burgard, W.; Fox, D.; Fröhlinghaus, T.; Hennig, D.; Hofmann, T.; Krell, M.; and Schmidt, T. 1998. Map learning and high-speed navigation in rhino. In Kortenkamp, D.; Bonasso, R.; and Murphy, R., editors 1998, *Artificial Intelligence Based Mobile Robotics: Case Studies of Successful Robot Systems.* MIT Press. 21–52.